# Modeling Interactions of the Rat's Place and Head Direction Systems

**A. David Redish and David S. Touretzky**
Computer Science Department & Center for the Neural Basis of Cognition
Carnegie Mellon University, Pittsburgh PA 15213-3891
Internet: {dredish,dst}@cs.cmu.edu

## Abstract

We have developed a computational theory of rodent navigation that includes analogs of the place cell system, the head direction system, and path integration. In this paper we present simulation results showing how interactions between the place and head direction systems can account for recent observations about hippocampal place cell responses to doubling and/or rotation of cue cards in a cylindrical arena (Sharp *et al.*, 1990).

Rodents have multiple internal representations of their relationship to their environment. They have, for example, a representation of their location (place cells in the hippocampal formation, see Muller *et al.*, 1991), and a location-independent representation of their heading (head direction cells in the postsubiculum and the anterior thalamic nuclei, see Taube *et al.*, 1990; Taube, 1995).

If these representations are to be used for navigation, they must be aligned consistently whenever the animal reenters a familiar environment. This process was examined in a set of experiments by Sharp *et al.* (1990).

## 1 The Sharp *et al.*, 1990 experiment

Rats spent multiple sessions finding food scattered randomly on the floor of a black cylindrical arena with a white cue card along the wall subtending 90° of arc. The animals were not disoriented before entering the arena, and they always entered at the same location: the northwest corner. See Figure 3a. Hippocampal place fields were mapped by single-cell recording. A variety of probe trials were then introduced. When an identical second cue

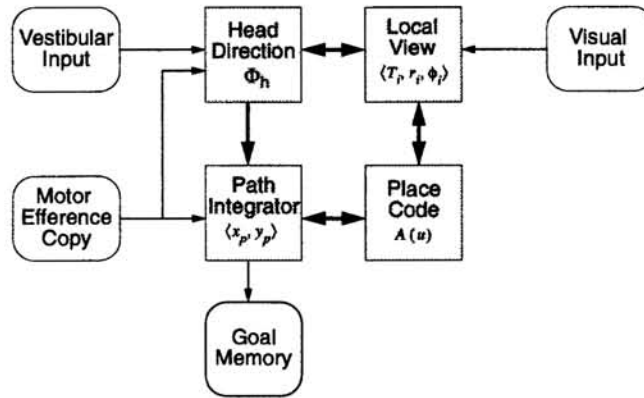

Figure 1: Organization of the rodent navigation model.

card was added opposite the first (Figure 3c), most place fields did not double.[1] Instead, the cells continued to fire at their original locations. However, if the rat was introduced into the double-card environment at the *southeast* corner (Figure 3d), the place fields rotated by 180°. But rotation did not occur in single-card probe trials with a southeast entry point (Figure 3b). When tested with cue cards rotated by ±30°, Sharp *et al.* observed that place field locations were controlled by an interaction of the choice of entry point with the cue card positions (Figure 3f.)

## 2    The CRAWL model

In earlier work (Wan *et al.*, 1994a; Wan *et al.*, 1994b; Redish and Touretzky, 1996) we described a model of rodent navigation that includes analogs of both place cells and the head direction system. This model also includes a local view module representing egocentric spatial information about landmarks, and a separate metric representation of location which serves as a substrate for path integration. The existence of a path integration faculty in rodents is strongly supported by behavioral data; see Maurer and Seguinot (1995) for a discussion. Hypotheses about the underyling neural mechanismss are presently being explored by several researchers, including us.

The structure of our model is shown in Figure 1. Visual inputs are represented as triples of form $\langle T_i, r_i, \theta_i \rangle$, each denoting the type, distance, and egocentric bearing of a landmark. The experiments reported here used two point-type landmarks representing the left and right edges of the cue card, and one surface-type landmark representing the arena wall. For the latter, $r_i$ and $\theta_i$ define the normal vector between the rat and the surface. In the local view module, egocentric bearings $\theta_i$ are converted to allocentric form $\phi_i$ by adding the current value represented in the head direction system, denoted as $\Phi_h$. The visual angle $\alpha_{ij}$ between pairs of landmarks is also part of the local view, and can be used to help localize the animal when its head direction is unknown. See Figure 2.

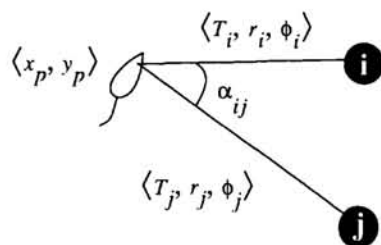

Figure 2: Spatial variables used in tuning a place cell to two landmarks $i$ and $j$ when the animal is at path integrator coordinates $\langle x_p, y_p \rangle$.

Our simulated place units are radial basis functions tuned to combinations of individual landmark bearings and distances, visual angles between landmark pairs, and path integrator coordinates. Place units can be driven by visual input alone when the animal is trying to localize itself upon initial entry at a random spot in the environment, or by the path integrator alone when navigating in the dark. But normally they are driven by both sources simultaneously. A key role of the place system is to maintain associations between the two representations, so that either can be reconstructed from the other. The place system also maintains a record of allocentric bearings of landmarks when viewed from the current position; this enables the local view module to compare perceived with remembered landmark bearings, so that drift in the head direction system can be detected and corrected.

In computer simulations using a single parameter set, the model reproduces a variety of behavioral and neurophysiological results including control of place fields by visual landmarks, persistence of place fields in the dark, and place fields drifting in synchrony with drift in the head direction system. Its predictions for open-field landmark-based navigation behavior match many of the experimental results of Collett *et al.* (1986) for gerbils.

## 2.1 Entering a familiar environment

Upon entering a familiar environment, the model's four spatial representations (local view, head direction, place code, and path integrator coordinates) must be aligned with the current sensory input and with each other. Note that local view information is completely determined given the visual input and head direction, and place cell activity is completely determined given the local view and path integrator representations. Thus, the alignment process manipulates just two variables: head direction and path integrator coordinates. When the animal enters the environment with initial estimates for them, the alignment process can produce four possible outcomes: (1) Retain the initial values of both variables, (2) Reset the head direction, (3) Reset the path integrator, or (4) Reset both head direction and the path integrator.

## 2.2 Prioritizing the outcomes

When the animal was placed at the northwest entry point and there were two cue cards (Figure 3c), we note that the orientation of the wall segment adjacent to the place field is identical with that in the training case. This suggests that the animal's head direction

did not change. The spatial relationship between the entry point and place field was also unchanged: notice that the distance from the entry point to the center of the field is the same as in Figure 3a. Therefore, we conclude that the initially estimated path integrator coordinates were retained. Alternatively, the animal could have changed both its head direction (by 180°) and its path integrator coordinates (to those of the southeast corner) and produced consistent results, but to the experimenter the place field would appear to have flipped to the other card. Because no flip was observed, the first outcome must have priority over the fourth.

In panel d, where the place field has flipped to the northwest corner, the orientation of the segment of wall adjacent to the field has changed, but the spatial relationship between the entry point and field center has not. Resetting the path integrator and not the head direction would also give a solution consistent with this local view, but with the place field unflipped (as in panel b). We conclude that the second outcome (reset head direction) must have priority over the third (reset the path integrator).

The third and fourth outcomes are demonstrated in Figures 3b and 3f. In panel b, the orientation of the wall adjacent to the place field is unchanged from panel a, but the spatial relationship between the entry point and the place field center is different, as evidenced by the fact that the distance between them is much reduced. This is outcome 3. In panel f, both variables have changed (outcome 4).

Finally, the fact that place fields are stable over an entire session, even when there are multiple cue cards (and therefore multiple consistent pairings of head directions and path integrator coordinates) implies that animals do not reset their head direction or path integrator in visually ambiguous environments as long as the current values are reasonably consistent with the local view. We therefore assume that outcome 1 is preferred over the others.

This analysis establishes a partial ordering over the four outcomes: 1 is preferred over 4 by Figure 3c, and over the others by the stability of place fields, and outcome 2 is preferred over 3 by Figure 3d. This leaves open the question of whether outcome 3 or 4 has priority over the other. In this experiment, after resetting the path integrator it's always safe for the animal to attempt to reset its head direction. If the head direction does not change by more than a few degrees, as in panel b, we observe outcome 3; if it does change substantially, as in panel f, we observe outcome 4.

## 2.3  Consistency

The viability of an outcome is a function of the *consistency* between the local view and path integrator representations. The place system maintains the association between the two representations and mediates the comparison between them.

The activity $A(u)$ of a place unit is the product of a local view term $LV(u)$ and a path integrator term $C(u)$. $LV(u)$ is in turn a product of five Gaussians: two tuned to bearings and two to distances (for the same pair of landmarks), and one tuned to the retinal angle between a pair of landmarks. $C(u)$ is a Gaussian tuned to the path integrator coordinates of the center of the place field.

If the two representations agree, then the place units activated by path integrator input will be the same as those activated by the local view module, so the product $A(u)$ computed by those units will be significantly greater than zero. The consistency $\kappa$ of the association

between path integrator and local view representations is given by: $\kappa = \sum_u A(u) / \sum_u C(u)$. Because $A(u) < C(u)$ for all place units, $\kappa$ ranges between 0 and 1. When the current local view is compatible with that predicted by the current path integrator coordinates, $\kappa$ will be high; when the two are not compatible, $\kappa$ will be low.

Earlier we showed that the navigation system should choose the highest priority viable outcome. If the consistency of an outcome is more than $\kappa^*$ better than all higher-priority outcomes, that outcome is a viable choice and higher-priority ones are not. $\kappa^*$ is an empirically derived constant that we have set equal to 0.04.

## 3   Discussion

Our results match all of the cases already discussed. (See Figure 3, panels a through d as well as f and h.) Sharp *et al.* (1990) did not actually test the rotated cue cards with a northwest entry point, so our result in panel e is a prediction.

When the animals entered from the northwest, but only one cue card was available at 180°, Sharp *et al.* report that the place field did not rotate. In our model the place field does rotate, as a result of outcome 4. This discrepancy can be explained by the fact that this particular manipulation was the last one in the sequence done by Sharp *et al.* McNaughton *et al.* (1994) and Knierim *et al.* (1995) have shown that if rats experience the cue card moving over a number of sessions, they eventually come to ignore it and it loses control over place fields. When we tested our model without a cue card (equivalent to a card being present but ignored), the resulting place field was more diffuse than normal but showed no rotation; see Figure 3g. We thus predict that if this experiment had been done before the other manipulations rather than after, the place field would have followed the cue card.

In the Sharp *et al.* experiment, the animals were always placed in the environment at the same location during training. Therefore, they could reliably estimate their initial path integrator coordinates. They also had a reliable head direction estimate because they were not disoriented. We predict that were the rats trained with a variety of entry points instead of just one, using an environment with a single cue card at 0° (the training environment used by Sharp *et al.*), and then tested with two cue cards at 0° and 180°, the place field would not rotate no matter what entry point was used. This is because when trained with a variable entry point, the animal would not learn to anticipate its path integrator coordinates upon entry; a path integrator reset would have to be done every time in order to establish the animal's coordinates. The reset mechanism uses allocentric bearing information derived from the head direction estimate, and in this task the resulting path integrator coordinates will be consistent with the initial head direction estimate. Hence, outcome 3 will always prevail.

If the animal is disoriented, however, then both the path integrator and the head direction system must be reset upon entry (because consistency will be low with a faulty head direction), and the animal must choose one cue card or the other to match against its memory. So with disorientation and a variable entry point, the place field will be controlled by one or the other cue card with a 50/50 probability. This was found to be true in a related behavioral experiment by Cheng (1986).

Our model shows how interactions between the place and head direction systems handle the various combinations of entry point, number of cue cards, and amount of cue card rotation. It predicts that head direction reset will be observed in certain tasks and not in others. In

experiments such as the single cue card task with an entry in the southeast, it predicts the place code will shift from an initial value corresponding to the northwest entry point to the value for the southeast entry point, but the head direction will *not* change. This could be tested by recording simultaneously from place cells and head direction cells.

## Footnotes

[1]Five of the 18 cells recorded by Sharp *et al.* changed their place fields over the various recording sessions. Our model does not reproduce these effects, since it does not address changes in place cell tuning. Such changes could occur due to variations in the animal's mental state from one trial to the next, or as a result of learning across trials.

# References

Cheng, K. (1986). A purely geometric module in the rat's spatial representation. *Cognition*, 23:149–178.

Collett, T., Cartwright, B. A., and Smith, B. A. (1986). Landmark learning and visuospatial memories in gerbils. *Journal of Comparative Physiology A*, 158:835–851.

Knierim, J. J., Kudrimoti, H. S., and McNaughton, B. L. (1995). Place cells, head direction cells, and the learning of landmark stability. *Journal of Neuroscience*, 15:1648–59.

Maurer, R. and Seguinot, V. (1995). What is modelling for? A critical review of the models of path integration. *Journal of Theoretical Biology*, 175:457–475.

McNaughton, B. L., Mizumori, S. J. Y., Barnes, C. A., Leonard, B. J., Marquis, M., and Green, E. J. (1994). Cortical rpresentation of motion during unrestrained spatial navigation in the rat. *Cerebral Cortex*, 4(1):27–39.

Muller, R. U., Kubie, J. L., Bostock, E. M., Taube, J. S., and Quirk, G. J. (1991). Spatial firing correlates of neurons in the hippocampal formation of freely moving rats. In Paillard, J., editor, *Brain and Space*, chapter 17, pages 296–333. Oxford University Press, New York.

Redish, A. D. and Touretzky, D. S. (1996). Navigating with landmarks: Computing goal locations from place codes. In Ikeuchi, K. and Veloso, M., editors, *Symbolic Visual Learning*. Oxford University Press. In press.

Sharp, P. E., Kubie, J. L., and Muller, R. U. (1990). Firing properties of hippocampal neurons in a visually symmetrical environment: Contributions of multiple sensory cues and mnemonic processes. *Journal of Neuroscience*, 10(9):3093–3105.

Taube, J. S. (1995). Head direction cells recorded in the anterior thalamic nuclei of freely moving rats. *Journal of Neuroscience*, 15(1):1953–1971.

Taube, J. S., Muller, R. I., and Ranck, Jr., J. B. (1990). Head direction cells recorded from the postsubiculum in freely moving rats. I. Description and quantitative analysis. *Journal of Neuroscience*, 10:420–435.

Wan, H. S., Touretzky, D. S., and Redish, A. D. (1994a). Computing goal locations from place codes. In *Proceedings of the 16th annual conference of the Cognitive Science society*, pages 922–927. Lawrence Earlbaum Associates, Hillsdale NJ.

Wan, H. S., Touretzky, D. S., and Redish, A. D. (1994b). Towards a computational theory of rat navigation. In Mozer, M., Smolensky, P., Touretzky, D., Elman, J., and Weigend, A., editors, *Proceedings of the 1993 Connectionist Models Summer School*, pages 11–19. Lawrence Earlbaum Associates, Hillsdale NJ.

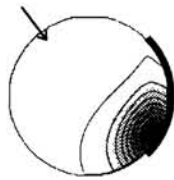

(a) 1 cue card at 0° (East)
entry in Northwest corner
angle of rotation (Sharp *et al.*) = 2.7°
precession of HD system = 0°

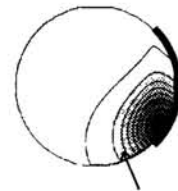

(b) 1 cue card at 0°
entry in Southeast corner
angle of rotation (Sharp *et al.*) = −6.0°
precession of HD system = 2°

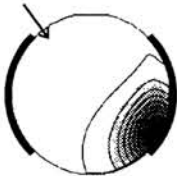

(c) 2 cue cards at 0° (East) & 180° (West)
entry in Northwest corner
angle of rotation (Sharp *et al.*) = −2.3°
precession of HD system = 0°

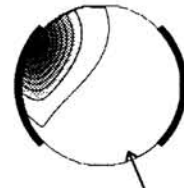

(d) 2 cue cards at 0° & 180°
entry in Southeast corner
angle of rotation (Sharp *et al.*) = 182.5°
precession of HD system = 178°

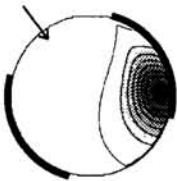

(e) 2 cue cards at 330° & 150°
entry in Northwest corner
*not done by Sharp* et al.
precession of HD system = 331°

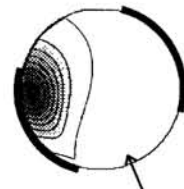

(f) 2 cue cards at 330° & 150°
entry in Southeast corner
angle of rotation (Sharp *et al.*) = 158.3°
precession of HD system = 151°

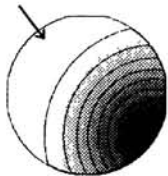

(g) 1 cue card at 180° (West)
entry in Northwest corner
angle of rotation (Sharp *et al.*) = −5.5°
precession of HD system = 0°

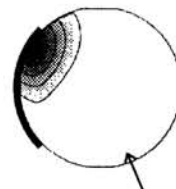

(h) 1 cue card at 180°
entry in Southeast corner
angle of rotation (Sharp *et al.*) = 182.2°
precession of HD system = 179°

Figure 3: Computer simulations of the Sharp *et al.* (1990) experiment showing that place fields are controlled by both cue cards (thick arcs) and entry point (arrowhead). "Angle of rotation" is the angle at which the correlation between the probe and training case place fields is maximal. Because head direction and place code are tightly coupled in our model, precession of HD is an equivalent measure in our model.